# Discrete Affine Wavelet Transforms For Analysis And Synthesis Of Feedforward Neural Networks

**Y. C. Pati** and **P. S. Krishnaprasad**
Systems Research Center and Department of Electrical Engineering
University of Maryland, College Park, MD 20742

## Abstract

In this paper we show that discrete affine wavelet transforms can provide a tool for the analysis and synthesis of standard feedforward neural networks. It is shown that wavelet frames for $L^2(\mathbb{R})$ can be constructed based upon sigmoids. The spatio-spectral localization property of wavelets can be exploited in defining the topology and determining the weights of a feedforward network. Training a network constructed using the synthesis procedure described here involves minimization of a convex cost functional and therefore avoids pitfalls inherent in standard backpropagation algorithms. Extension of these methods to $L^2(\mathbb{R}^N)$ is also discussed.

## 1  INTRODUCTION

Feedforward type neural network models constructed from empirical data have been found to display significant predictive power [6]. Mathematical justification in support of such predictive power may be drawn from various density and approximation theorems [1, 2, 5]. Typically this latter work doesn't take into account the spectral features apparent in the data. In the present paper, we note that the discrete affine wavelet transform provides a natural framework for the analysis and synthesis of feedforward networks. This new tool takes account of spatial and spectral localization properties present in the data.

Throughout most of this paper we restrict discussion to networks designed to approximate mappings in $L^2(\mathbb{R})$ . Extensions to $L^2(\mathbb{R}^N)$ are briefly discussed in Section 4 and will be further developed in [10].

## 2    WAVELETS AND FRAMES

Consider a function $f$ of one real variable as a static feedforward input-output map

$$y = f(x)$$

For simplicity assume $f \in L^2(\mathbb{R})$ the space of square integrable functions on the real line. Suppose a sequence $\{f_n\} \subset L^2(\mathbb{R})$ is given such that, for suitable constants $A > 0$, $B < \infty$,

$$A\|f\|^2 \le \sum_n |< f, f_n >|^2 \le B\|f\|^2 \tag{1}$$

for *all* $f \in L^2(\mathbb{R})$ . Such a sequence is said to be a *frame*. In particular orthonormal bases are frames. The above definition (1) also applies in the general Hilbert space setting with the appropriate inner product. Let $T$ denote the bounded operator from $L^2(\mathbb{R})$ to $l^2(\mathbb{Z})$, the space of square summable sequences, defined by

$$(Tf) = \{< f, f_n >\}_{n \in \mathbb{Z}}.$$

In terms of the *frame operator $T$*, it is possible to give series expansions,

$$
\begin{aligned}
f &= \sum_n \tilde{f}_n < f, f_n > \\
  &= \sum_n f_n < f, \widetilde{f_n} >,
\end{aligned}
\tag{2}
$$

where $\{\tilde{f}_n = (T^*T)^{-1}f_n\}$ is the *dual frame*.

A particular class of frames leads to affine wavelet expansions. Consider a family of functions $\{\psi_{mn}\}$ of the form,

$$\psi_{mn}(x) = a^{-m/2}\psi(a^{-m}x - nb) \tag{3}$$

where, the function $\psi$ satisfies appropriate admissibility conditions [3, 4] (e.g. $\int \psi = 0$). Then for suitable choices of $a > 1$, $b > 0$, the family $\{\psi_{mn}\}$ is a frame for $L^2(\mathbb{R})$ . Hence there exists a convergent series representation,

$$
\begin{aligned}
f(x) &= \sum_m \sum_n c_{mn} \psi_{mn}(x) \\
     &= \sum_m \sum_n c_{mn} a^{-m/2}\psi(a^{-m}x - nb)
\end{aligned}
\tag{4}
$$

The frame condition (1) guarantees that the operator $(T^*T)$ is boundedly invertible. Also since $\|I - (2(A+B)^{-1}T^*T)\| < 1$, $(T^*T)^{-1}$ is given by a Neumann series [3]. Hence, given $f$, the expansion coefficients $c_{mn}$ can be computed.

The representation (4) of $f$ above as a series in *dilations* and *translations* of a single function $\psi$ is called a wavelet expansion and the function $\psi$ is known as the *analyzing* or *mother wavelet* for the expansion.

# 3 FEEDFORWARD NETWORKS AND WAVELET EXPANSIONS

Consider the input-output relationship of a feedforward network with one input, one output, and a single hidden layer,

$$\widetilde{f}(x) = \sum_n c_n g(a_n x - b_n) \tag{5}$$

where $a_n$ are the weights from the the input node to the hidden layer, $b_n$ are the biases on the hidden layer nodes, $c_n$ are the weights from the hidden layer to the output layer and $g$ defines the activation function of the hidden layer nodes. It is clear from (5) that the output of such a network is given in terms of dilations and translations of a single function $g$.

## 3.1 WAVELET ANALYSIS OF FEEDFORWARD NETWORKS

Let $g$ be a 'sigmoidal' function e.g. $g(x) = \frac{1}{1+e^{-x}}$ and let $\psi$ be defined as

$$\psi(x) = g(x+2) + g(x-2) - 2g(x). \tag{6}$$

Then it is possible (see [9] for details) to determine a translation stepsize $b$ and

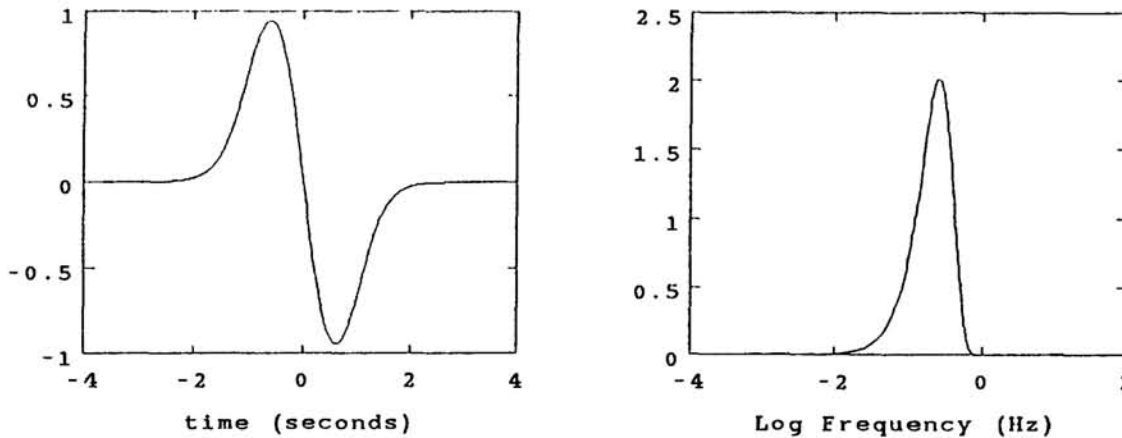

Figure 1: Mother Wavelet $\psi$ (Left) And Magnitude Of Fourier Transform $|\widehat{\psi}|^2$

a dilation stepsize $a$ for which the family of functions $\psi_{mn}$ as defined by (3) is a frame for $L^2(\mathbb{R})$ . Note that wavelet frames for $L^2(\mathbb{R})$ can be constructed based upon other combinations of sigmoids (e.g $\psi(x) = g(x+p) + g(x-p) - 2g(x)$, $p > 0$) and that we use the mother wavelet of (6) only to illustrate some properties which are common to many such combinations.

It follows from the above discussion that a feedforward network having one hidden layer with sigmoidal activation functions can represent any function in $L^2(\mathbb{R})$ . In such a network (6) says that the sigmoidal nodes should be grouped together in sets of three so as to form the mother wavelet $\psi$.

## 3.2    WAVELETS AND SYNTHESIS OF FEEDFORWARD NETWORKS

In defining the topology of a feedforward network we make use of the fact that the function $\psi$ is well concentrated in both spatial and spectral domains (see Figure 1). Dilating $\psi$ corresponds to shifting the spectral concentration and translating $\psi$ corresponds to shifting the spatial concentration.

The synthesis procedure we describe here is based upon estimates of the spatial and spectral localization of the unknown mapping as determined from samples provided by the training data. Spatial locality of interest can easily be determined by examination of the training data or by introducing *a priori* assumptions as to the region over which it is desired to approximate the unknown mapping. Estimates of the appropriate spectral locality are also possible via preprocessing of the training data.

Let $\mathcal{Q}_{mn}$ and $\mathcal{Q}_f$ respectively denote the spatio-spectral concentrations of the wavelet $\psi_{mn}$ and of $f$. Thus $\mathcal{Q}_{mn}$ and $\mathcal{Q}_f$ are rectangular regions in the spatio-spectral plane (see Figure 2) which contain 'most' of the energy in the functions $\psi_{mn}$ and $f$. More precise definitions of these concentrations can be found in [9]. Assuming that $\mathcal{Q}_f$ has been estimated from the training data. We choose only those

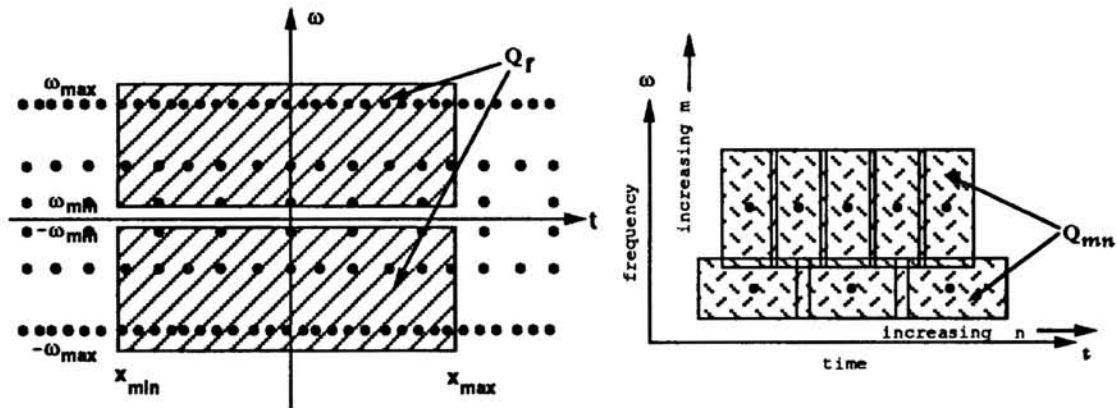

Figure 2: Spatio-Spectral Concentrations $\mathcal{Q}_{mn}$ And $\mathcal{Q}_f$ Of Wavelets $\psi_{mn}$ And Unknown Map $f$.

elements of the frame $\{\psi_{mn}\}$ which contribute 'significantly' to the region $\mathcal{Q}_f$ by defining an index set $\mathcal{I}_f \subseteq \mathbf{Z}^2$ in the following manner,

$$\mathcal{I}_f = \{(m,n) \in \mathbf{Z} : \mu(\mathcal{Q}_f \bigcap \mathcal{Q}_{mn}) > 0\}$$

where, $\mu$ is the Lesbegue measure on $\mathbb{R}^2$. Since $f$ is concentrated in $\mathcal{Q}_f$, by choosing $\mathcal{I}_f$ as above, a 'good' approximation of $f$ can be obtained in terms of the *finite* set of frame elements with indices in $\mathcal{I}_f$. That is $f$ can be approximated by $\tilde{f}$ where,

$$\tilde{f} = \sum_{(m,n) \in \mathcal{I}_f} c_{mn} \psi_{mn} \qquad (7)$$

for some coefficients $\{c_{mn}\}_{(m,n)\in\mathcal{I}_f}$.

Having determined $\mathcal{I}_f$, a network is constructed to implement the appropriate wavelets $\psi_{mn}$. This is easily accomplished by choosing the number of sigmoidal hidden layer nodes to be $M = 3 \times \sharp\mathcal{I}_f$ and then grouping them together in sets of three to implement $\psi$ as in (6). Weights from the input to the hidden layer are set to provide the required dilations of $\psi$ and biases on the hidden layer nodes are set to provide the required translations.

### 3.2.1 Computation of Coefficients

By the above construction, all weights in the network have been fixed except for the weights from the hidden layer to the output which specify the coefficients $c_{mn}$ in (7). These coefficients can be computed using a simple gradient descent algorithm on the standard cost function of backpropagation. Since the cost function is convex in the remaining weights, only globally minimizing solutions exist.

### 3.2.2 Simulations

Figure 3 shows the results of a simple simulation example. The solid line in Figure 3 indicates the original mapping $f$ which was defined via the inverse Fourier transform of a randomly generated approximately bandlimited spectrum. Using a single dilation of $\psi$ which covered the frequency band sufficiently well and the required translations, the dashed curve shows the learned network approximation.

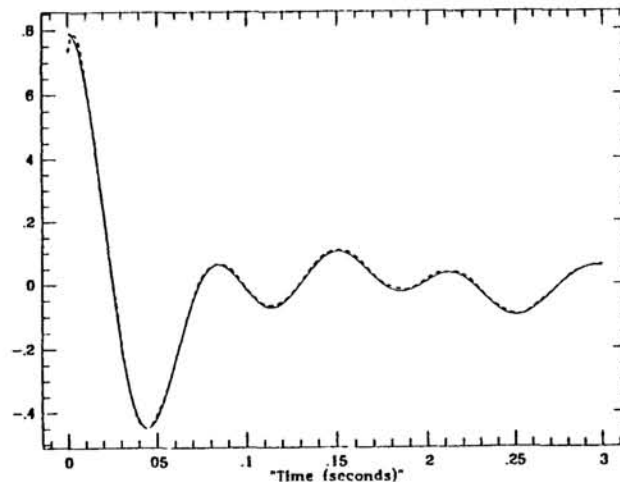

Figure 3: Simulation Using Network Synthesis Procedure. Solid Curve: Original Function, Dashed Curve: Network Reconstruction.

## 4 DISCUSSION AND CONCLUSIONS

It has been demonstrated here that affine wavelet expansions provide a framework within which feedforward networks designed to approximate mappings in $L^2(\mathbb{R})$ can be understood. In the case when the mapping is known, the expansion coefficients, and therefore *all* weights in the network can be computed. Hence the wavelet

transform method (and in general any transform method) not only gives us representability of certain classes of mappings by feedforward networks, but also tells us what the representation should be. Herein lies an essential difference between the wavelet methods discussed here and arguments based upon density in function spaces.

In addition to providing arguments in support of the approximating power of feedforward networks, the wavelet framework also suggests one method of choosing network topology (in this case the number of hidden layer nodes) and reducing the training problem to a convex optimization problem. The synthesis technique suggested is based upon spatial and spectral localization which is provided by the wavelet transform.

Most useful applications of feedforward networks involve the approximation of mappings with higher dimensional domains e.g. mappings in $L^2(\mathbb{R}^N)$. Discrete affine wavelet transforms can be applied in higher dimensions as well (see e.g. [7] and [8]). Wavelet transforms in $L^2(\mathbb{R}^N)$ can also be defined with respect to mother wavelets constructed from sigmoids combined in a manner which doesn't deviate from standard feedforward network architectures [10]. Figure 4 shows a mother wavelet for $L^2(\mathbb{R}^2)$ constructed from sigmoids. In higher dimensions it is possible to use more than one analyzing wavelet [7], each having certain orientation selectivity in addition to spatial and spectral localization. If orientation selectivity is not essential, an isotropic wavelet such as that in Figure 4 can be used.

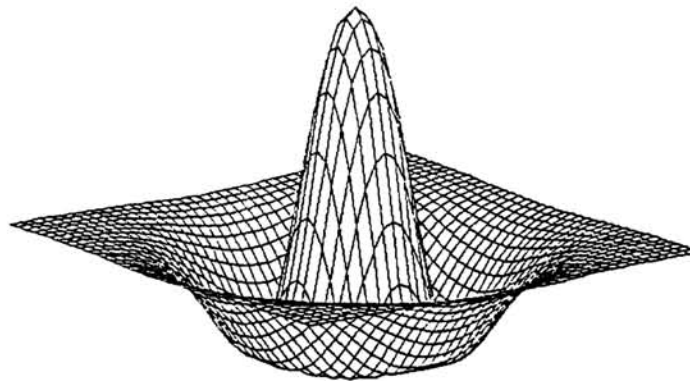

Figure 4: Two-Dimensional Isotropic Wavelet From Sigmoids

The wavelet formulation of this paper can also be used to generate an orthonormal basis of compactly supported wavelets within a standard feedforward network architecture. If the sigmoidal function $g$ in Equation (6) is chosen as a discontinuous threshold function, the resulting wavelet $\psi$ is the Haar function which thereby results in the Haar transform. Dilations of the Haar function in powers of 2 ($a = 2$) together with integer translations ($b = 1$), generate an orthonormal basis for $L^2(\mathbb{R})$ . Multidimensional Haar functions are defined similarly. The Haar transform is the earliest known example of a wavelet transform which however suffers due to the discontinuous nature of the mother wavelet.

## Acknowledgements

The authors wish to thank Professor Hans Feichtinger of the University of Vienna, and Professor John Benedetto of the University of Maryland for many valuable discussions. This research was supported in part by the National Science Foundation's Engineering Research Centers Program: NSFD CDR 8803012, the Air Force Office of Scientific Research under contract AFOSR-88-0204 and by the Naval Research Laboratory.

## References

[1] G. Cybenko. *Approximations by Superpositions of a Sigmoidal Function.* Technical Report CSRD 856, Center for Supercomputing Research and Development, University of Illinois, Urbana, February 1989.

[2] G. Cybenko. *Continuous Valued Neural Networks with Two Hidden Layers are Sufficient.* Technical Report, Department of Computer Science, Tufts University, Medford, MA, March 1988.

[3] I. Daubechies. The Wavelet Transform, Time-Frequency Localization and Signal Analysis. *IEEE Transactions on Information Theory*, 36(5):961–1005, September 1990.

[4] C. E. Heil and D. F. Walnut. Continuous and Discrete Wavelet Transforms. *SIAM Review*, 31(4):628–666, December 1989.

[5] K. Hornik, M. Stinchcombe, and H. White. Multilayer Feedforward Networks are Universal Approximators. *Neural Networks*, 2:359–366, 1989.

[6] A. Lapedes, and R. Farber. *Nonlinear Signal Processing Using Neural Networks: Prediction and System Modeling.* Technical Report LA-UR-87-2662, Los Alamos National Laboratory, 1987.

[7] S. G. Mallat. Multifrequency Channel Decompositions of Images and Wavelet Models. *IEEE Transactions On Acoustics Speech and Signal Processing*, 37(12):2091–2110, December 1989.

[8] R. Murenzi, "Wavelet Transforms Associated To The n-Dimensional Euclidean Group With Dilations: Signals In More Than One Dimension," in *Wavelets Time-Frequency Methods And Phase Space* (J. M. Combes, A. Grossman and Ph. Tchamitchian, eds.), pp. 239–246, Springer-Verlag, 1989.

[9] Y. C. Pati and P. S. Krishnaprasad, "Analysis and Synthesis of Feedforward Neural Networks Using Discrete Affine Wavelet Transforms," Technical Report SRC TR 90-44, University of Maryland, Systems Research Center, 1990.

[10] Y. C. Pati and P. S. Krishnaprasad, In preparation.